# VeXKD: The Versatile Integration of Cross-Modal Fusion and Knowledge Distillation for 3D Perception

**Yuzhe JI**[1], **Yijie Chen**[1], **Liuqing Yang**[1,2], **Rui Ding**[3], **Meng Yang**[3], **Xinhu Zheng**[†1,2]

[1] Hong Kong University of Science and Technology (Guangzhou), Guangzhou, China
[2] Hong Kong University of Science and Technology, Hong Kong SAR, China
[3] Xi'an Jiaotong University, Xi'an, China
{yji755, ychen324}@connect.hkust-gz.edu.cn
lqyang@hkust-gz.edu.cn   dingrui@stu.xjtu.edu.cn
mengyang@xjtu.edu.cn   xinhuzheng@hkust-gz.edu.cn

## Abstract

Recent advancements in 3D perception have led to a proliferation of network architectures, particularly those involving multi-modal fusion algorithms. While these fusion algorithms improve accuracy, their complexity often impedes real-time performance. This paper introduces **VeXKD**, an effective and **Ve**rsatile framework that integrates **Cross**-Modal Fusion with **K**nowledge **D**istillation. VeXKD applies knowledge distillation exclusively to the Bird's Eye View (BEV) feature maps, enabling the transfer of cross-modal insights to single-modal students without additional inference time overhead. It avoids volatile components that can vary across various 3D perception tasks and student modalities, thus improving versatility. The framework adopts a modality-general cross-modal fusion module to bridge the modality gap between the multi-modal teachers and single-modal students. Furthermore, leveraging byproducts generated during fusion, our BEV query guided mask generation network identifies crucial spatial locations across different BEV feature maps from different tasks and semantic levels in a data-driven manner, significantly enhancing the effectiveness of knowledge distillation. Extensive experiments on the nuScenes dataset demonstrate notable improvements, with up to 6.9%/4.2% increase in mAP and NDS for 3D detection tasks and up to 4.3% rise in mIoU for BEV map segmentation tasks, narrowing the performance gap with multi-modal models.

## 1 Introduction

3D perception, encompassing 3D object detection [31, 1] and BEV map segmentation [63, 22, 72], is crucial for understanding 3D scenes and controlling autonomous vehicles [48, 6, 38]. Achieving high accuracy and real-time performance simultaneously presents the desirable yet challenging pursuit in this field [12]. Many studies [2, 40, 28] focus on multi-modal fusion, particularly the fusion of LiDAR and multi-view cameras, to enhance perception accuracy. These methods necessitate handling additional input data, inevitably employ more complex networks, and extend inference time. Conversely, some studies [69, 52, 66, 65, 60] continue to focus on single-modal algorithms to maintain system simplicity and enhance accuracy with better training strategies and data pipelines.

Cross-modal Knowledge Distillation (KD) [14, 36, 59] has emerged as a promising strategy for transferring insights across modalities, using multi-modal models as teachers and single-modal models as students. Cross-modal KD can thus improve the accuracy of student models without incurring

---

[†]Corresponding authors.

additional inference time. However, cross-modal KD faces significant challenges, including capacity discrepancies between the student and teacher models, and information gaps stemming from the different input modalities. Xue *et al.* [50] propose and tentatively validate that teacher models focusing on modality-specific information can exacerbate the modality gap between student, reducing the effectiveness of cross-modal KD. In contrast, teacher models that base decisions on modality-general information can minimize disparities with single-modal students, facilitating a more effective cross-modal KD process.

However, the current research paradigm, which separates cross-modal fusion from KD, limits its potential synergies. On one hand, multimodal fusion research [24, 62, 28] has become performance-oriented. Despite achieving impressive results on benchmarks like nuScenes [4], BEVFusion [28] stands out as the state-of-the-art fusion method. However, the visualization of fusion and LiDAR feature maps in BEVFusion [28], and the significant performance drops under conditions like LiDAR failure as depicted in Fig. 1, reveal an over-reliance on LiDAR-specific information rather than on the general information of multi-modal features. This makes these high performance

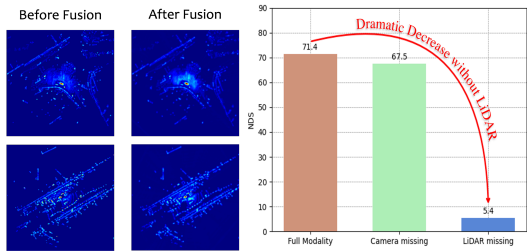

Figure 1: (Left) Visualization of BEVFusion feature maps before and after fusion, indicating minimal information gain. (Right) Significant performance degradation in the LiDAR-missing scenario, highlighting the over-reliance on LiDAR features.

fusion methods less suitable as teachers in cross-modal KD. On the other hand, research on cross-modal KD in 3D perception [68, 7] seldom considers the selection of teachers, often directly using a LiDAR or fusion model with higher accuracy, such as BEVFusion [28], as the teacher [68], which potentially limits the effectiveness of cross-modal KD.

The versatility of KD algorithms is another facet often overlooked in previous research, yet it significantly impacts the vitality and breadth of KD applicability, especially in a rapidly evolving field like 3D perception. If a KD algorithm cannot be directly applied to new student algorithms, its re-implementation may require numerous difficult-to-generalize empirical engineering decisions and result in an unpredictable performance drop. The versatility of a KD algorithm primarily hinges on two aspects. Firstly, it depends on whether the KD algorithm is tied to the processing steps or feature space of a specific modality. For instance, MonoDistill [9] projects LiDAR points onto a perspective view and uses the same model architecture as the camera, serving as depth supervision for camera branches but is not applicable to LiDAR students. In contrast, using the BEV space, which is representation-friendly to different modalities, alleviates this limitation. Secondly, and most importantly, the versatility of a KD algorithm depends on its compatibility with specific network architectures, especially task-specific heads, which exhibit the greatest design variations across different 3D perception algorithms and downstream tasks. These can range from dense heads [55, 33, 37, 34, 30] and transformer-based heads [2, 45, 58, 27] to segmentation heads [72, 46, 49, 67]. This diversity limits the versatility of KD methods that rely on response distillation across different perception tasks and various detection algorithms. Consequently, previous 3D perception KD methods often require identical heads and similar model architectures for both student and teacher models, and are generally limited to a single downstream task, primarily 3D object detection.

The current reliance on response distillation primarily stems from the underutilization of the rich information available in the teacher's feature maps. Previous efforts have focused on the ground truth locations, using Gaussian masks [68, 7] or sampling a few points within these areas [68]. These methods tend to overlook the valuable background information, which has proven useful in 2D perception [13, 61]. Moreover, by confining feature distillation solely to ground truth locations, these methods not only fail to account for variations in spatial perceptual fields across different levels of BEV feature maps but also struggle to generalize to tasks requiring dense supervision, such as map segmentation. This reduce the versatility of their KD methods.

To address the aforementioned challenges, we propose an effective and **ve**rsatile framework that integrates **Cross**-Modality Fusion and **K**nowledge **D**istillation within the BEV feature space. This framework effectively narrows the modality gap between the teacher and different single-modal student models by training a modality-general fusion teacher. Moreover, we leverage the learned BEV query, the byproduct of the fusion process, to guide a mask generation network in creating unique

spatial masks for different feature levels and tasks in a data-driven manner. These learned masks significantly enhance the effectiveness of feature distillation by enabling the selective filtration and transfer of valuable information from the teacher's feature maps. Furthermore, our KD framework is independent of specific processing steps or network architectures, making it versatile to be applied across any downstream 3D perception task and adaptable to various student modalities, as well as future advancements in this field.

In summary, our contributions are summarized as follows:

- We present an early effort to integrate cross-modal fusion and KD in 3D perception, enhancing the efficacy of cross-modal KD with the modality-general fusion teacher and the fusion byproduct.
- We pioneer data-driven spatial masks learning for feature distillation on BEV feature maps, selectively transferring only beneficial feature information by applying distinct masks tailored to different BEV feature maps.
- Our KD approach is designed to be task- and modality-agnostic, making it highly versatile for any BEV-based 3D perception task and adaptable to various student modalities.

## 2    Related Work

### 2.1    LiDAR-camera Fusion

LiDAR and camera are the two most common sensors used in 3D perception. Recent advancements in both camera-based [32, 16, 23, 25, 53] and LiDAR-based [69, 51, 19, 55] methods have achieved notable results, providing a solid foundation for LiDAR-camera fusion. Since the BEV space offers a feature space that is friendly to both modalities and can easily be applied to various downstream tasks, fusion research based on BEV space has also become a trend [26, 24, 42, 28]. In BEV-based perception methods, both single-modal and fusion models [32, 25, 19, 28] initially transform inputs from LiDAR or cameras into the BEV space using modality-specific encoders. Then only the fusion models being integrated through a fusion module, but finally a BEV encoder and task-specific heads are applied to all models for perception tasks. Although the abundance of research related to BEV poses challenges in selecting appropriate student and teacher architectures for KD, the common paradigm in BEV-based work offers the potential for a unified KD framework.

### 2.2    Knowledge Distillation

Knowledge distillation (KD) facilitates the efficient transfer of implicit knowledge from the teacher to the student without increasing the student's inference time [3, 14]. Besides aligning the teacher's soft outputs with the student's, some works indicate that mimicking the feature map can also boost performance [41, 21, 43, 56]. Direct distillation over the entire feature maps can potentially degrade performance due to the noisy teacher feature maps. Based on this observation, some approaches have resorted to attentive distillation, concentrating KD on the less noisy foreground features and using ground truth as a mask to guide the process [35, 57]. On the other hand, recent studies [13, 61] emphasize the value of background information and suggest decoupled KD processes for both foreground and background features. DistillBEV [47] advances this concept by decomposing the regions of feature maps and enhance attention to false positive regions. Recently, inspired by pretext tasks in large language models, generative distillation has been proposed [54]. However, random masks utilized in generative distillation can destabilize the performance, especially in 3D object detection with pronounced foreground-background imbalance. Our approach still aligns more closely with attentive distillation to selectively transfer knowledge from teacher's feature map.

Another key research focus is the efficacy of KD, as explored by Cho *et al.* [8]. Studies indicate that a high-accuracy teacher model does not necessarily improve KD results. Similarly, in cross-modal KD scenarios, Xue *et al.* [50] investigate factors influencing KD's effectiveness. They reveal that if a teacher makes decisions based on modality-general features, KD performance can be improved even when the teacher's accuracy is not superior. Recently, Huang *et al.* [17] develop a "vision-centric" multi-modal teacher, reducing reliance on LiDAR to align more closely with camera-based students. Our work aims to develop a modality-general fusion model without modifying the pipeline of the teacher network.

Due to spatial inaccuracies in RGB images, numerous studies have used KD to improve the accuracy of camera models. BEV-LGKD [20] uses LiDAR to enhance the camera-based students' depth estimation but it is limited to LSS-based [32] projection methods, excluding certain camera students like BEVFormer [25]. BEVDistill [7] introduces a sparse instance distillation method coupled with transformer-based detection heads, which is unsuitable for common dense detection heads [55]. Compared to camera student models, cross-modal KD applied to LiDAR students is relatively less explored. Zheng *et al.* [64] employ the PointPainting [39] model as a teacher, providing supervision only for voxel-based LiDAR students during the voxelization process. Unidistill [68] supports various student-teacher modality combinations, but its response distillation is limited to dense detection heads, and feature distillation depends on 3D detection ground truths, restricting its use in tasks like BEV map segmentation. To date, no universal KD paradigm covers diverse 3D perception tasks and student modalities.

# 3  Methodology

## 3.1  Overall architecture of Cross-Modal Fusion and Knowledge Distillation Framework

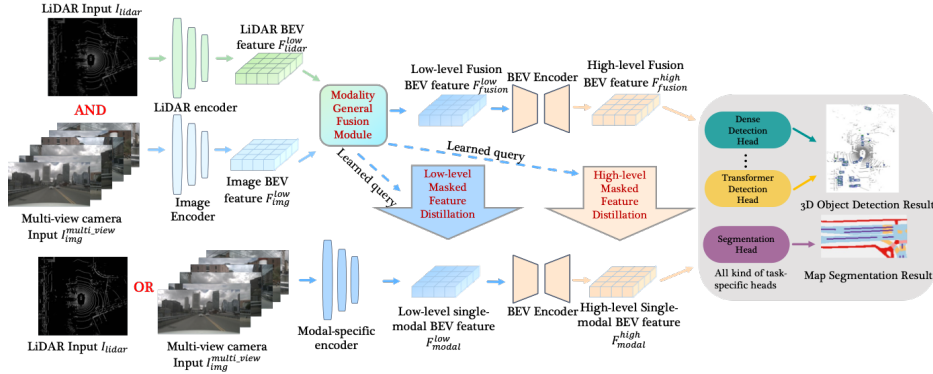

Figure 2: **Overall architecture of VeXKD**: Building upon the common BEV fusion pipeline, we tailor a Modality-General Fusion Module and design a masked feature distillation method with learned masks assisted by the byproduct of the fusion module, applied across both low-level and high-level BEV features. Our feature distillation framework circumvents variations in different model architectures, making it modality- and task-agnostic.

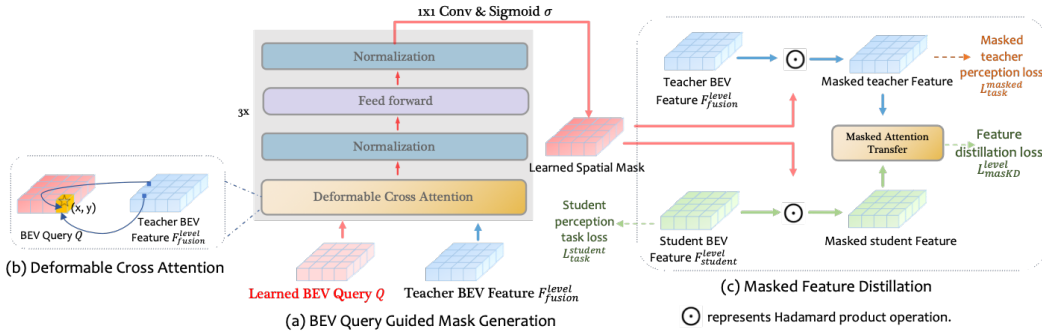

Figure 3: **Illustration of BEV query guided mask generation and masked feature distillation.** (a) Overview of the transformer-based block for mask generation, which adopts the byproducts from the fusion module as the BEV query. (b) In the deformable cross attention operation, the BEV query interacts with the teacher feature maps to identify crucial spatial locations. (c) In the masked feature distillation stage, learned spatial masks are applied to both teacher and student feature maps before calculating the distillation loss.

The overall architecture of the proposed framework is depicted in Fig. 2. The modality-general fusion module operates on the low-level BEV feature $F_{modal}^{low}$, capturing high spatial granularity and rich semantic information from different modalities. Enhanced by the byproducts of the fusion

module, masked feature distillation is applied to both low and high-level BEV features to selectively mimic the semantic information from the feature maps of the fusion teacher.

## 3.2 Modality-General Fusion Module (MGFM)

The architecture of the fusion module is shown in Fig. 4. Inspired by BEVFormer [25], we employ deformable attention [71] as the central mechanism of our fusion module. As detailed in Eq. 1, $q, p, x$ represent the query, reference point, and input features, respectively. The number of attention heads and the number of key points sampled per head are denoted by $N_{head}$ and $N_{key}$. $\mathcal{W}'_i$, and $\mathcal{W}_i$ are learnable parameters. $A_{ij}$ and $\Delta p_{ij} \in \mathbb{R}^2$ represent the predicted weight and the offset relative to the reference point $p$ for these sampled key points, both are learned from $q$. Deformable attention enables learnable sampling offsets, which provide a larger and more adaptable receptive field while maintaining computational efficiency. It is ideal

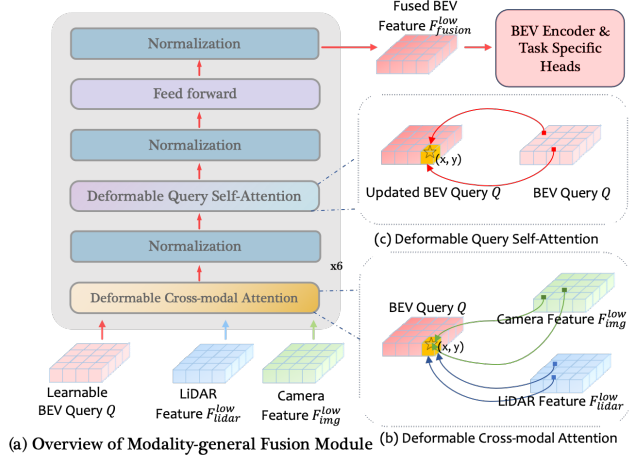

Figure 4: **The architecture of the Modality-General Fusion Module**: (a) Overview of the transformer-based block. (b) Deformable cross-modal attention operation: the BEV query symmetrically interacts with features sampled from both LiDAR and camera. (c) Deformable query self-attention operation: the BEV query interacts with itself to integrate correlational relationships.

for correcting varying degrees of spatial misalignments caused by ill-posed view projections and fickle sensor extrinsic matrices. Our fusion module adheres to the traditional transformer block structure, incorporating deformable cross-modal attention and query self-attention operations to iteratively fuse multi-modal BEV features.

During the deformable cross-modal attention operation, a set of learnable dense BEV queries $Q \in \mathbb{R}^{(H \times W) \times C}$ is initialized. These queries interact with low-level feature maps from each modality $F^{low}_{modal}$, for $modal \in \{lidar, camera\}$. The sampling offset generation process is modified by concatenating $F^{low}_{modal}$ with the query $Q$ to produce modality-specific sampling offsets $\Delta p$ and attention weights $\mathcal{W}$. The resulting feature map is updated by summing the outputs, as denoted in Eq. 2.

This symmetrical and equitable fusion approach forces the preservation of the information from each modality. It promotes the extraction of modality-general information from the fusion features, rather than disproportionately relying on a specific modality, thereby narrowing the information gap between the fusion teacher and individual modalities. In the deformable query self-attention operation, modality-specific feature inputs are substituted with the query $Q$ itself, as indicated in Eq. 3. This self-attention operation not only expands the receptive field but also capture the inter-correlations among BEV features, making the fusion features geometrically more accurate and semantically richer. After stacking 6 transformer blocks similar to BEVFormer[25], the MGFM module yields the fusion result $F^{low}_{fusion}$.

$$\text{DeformAttn}(q, p, x) = \sum_{i=1}^{N_{head}} \mathcal{W}_i \sum_{j=1}^{N_{key}} \mathcal{A}_{ij} \cdot \mathcal{W}'_i x(p + \Delta p_{ij}) \tag{1}$$

$$Q'_{modal} = [Q; F^{low}_{modal}]; \quad Q = \sum_{modal} \text{DeformAttn}\left(Q'_{modal}, p, F^{low}_{modal}\right) \tag{2}$$

$$Q = \text{DeformAttn}\left(Q, p, Q\right) \tag{3}$$

### 3.3 BEV Query Guided Mask Generation

Due to the inherent differences among various downstream tasks and feature levels, the specific areas within the teacher feature map that require the student's focus can vary. For example, in detection tasks, information near ground truth positions is crucial for the high-level BEV features. However, since high-level features selectively aggregate a broader range of low-level features, important low-level features may be more widely distributed, not just at ground truth positions. Based on these insights, we opt to learn ad hoc masks in a data-driven manner. Nevertheless, efficiently learning these masks rapidly necessitates a set of initial parameters that are easy to generalize. Once trained within the fusion module, the learned BEV query $Q$ can effectively extract valuable information from the BEV feature maps, facilitating quick adaptation to ad hoc spatial masks. To maximize the effectiveness of $Q$, we employ the deformable cross-attention operation in the mask generation network, similar to the setup in the fusion module as depicted in Fig. 3(a). Here, $Q$ and the teacher BEV feature map $F_{level}^{fusion}$, where $level \in \{low, high\}$, serve as the $q$ and $x$ for DeformAttn, respectively, allowing for continuous updates to $Q^{level}$. Finally, the channel dimension is reduced to 1 using a $1 \times 1$ convolution, and the values interval for the learned spatial mask $M_{level}$ is adjusted to range (0,1) through a sigmoid activation function.

In designing the mask generation loss, inspired by [18], the original teacher feature map $F_{fusion}^{level}$ is substituted with the masked version $\hat{F}_{fusion}^{level}$, obtained by applying Hadamard product operation $\odot$ between the teacher feature and the learned spatial mask, as shown in Eq. 4. This masked feature serves as input to derive the final masked teacher loss for downstream tasks $L_{masked}^{task}$. Minimizing this loss ensures the preservation of valuable features in the teacher feature map through the learned mask.

However, using only $L_{masked}^{task}$ as the mask learning loss can result in an all-ones mask. We also incorporate the feature distillation loss $\mathcal{L}_{masKD}$ into the objective to stabilize the mask generation process. This inclusion allows the student feature map to participate, helping avoid optimization pitfalls that arise from hard-to-mimic locations due to modality gaps. The final mask generation loss $L_{mask\_gen}$ is depicted in Eq. 5, where $\mu$ is a factor to balance the scale of the two losses.

$$\hat{F}_{fusion}^{level} = M^{level} \odot F_{fusion}^{level}, \ level \in \{low, high\} \tag{4}$$

$$\mathcal{L}_{mask\_gen} = \mathcal{L}_{masked}^{task} + \mu\mathcal{L}_{masKD} \tag{5}$$

### 3.4 Masked Feature Distillation

After applying the learned mask to both the student and teacher feature maps as depicted in Eq. 6 and Fig. 3(c), various loss functions can be applied to quantify the discrepancies between them. Finally, Attention Transfer [56] is adopted to compute the KD loss between the student and teacher. This method effectively mitigates the adverse impacts of channel-wise heterogeneity that arise from different architectures. By applying L2-norm in Attention Transfer as shown in Eq. 7, greater emphasis is placed on spatial locations with higher activations or more discriminative features, thereby enhancing the robustness and effectiveness of feature distillation.

We apply mask generation and attention transfer across both low- and high-level BEV features. The mask generation process, which also incorporates $\mathcal{L}_{masKD}$, runs concurrently with the KD process. It can be halted after a certain number of epochs using a controllable hook. The overall loss for the student model $\mathcal{L}_{overall}^{student}$ (see Eq. 8) combines the downstream task loss $\mathcal{L}_{task}^{student}$ with KD loss $\mathcal{L}_{masKD}$, using $\lambda$ to balance the magnitude of these losses.

$$Q^{level}(\mathbf{F}) = \text{vec}\left(\mathcal{F}_{sum}\left(M^{level} \odot \mathbf{F}\right)\right) \tag{6}$$

$$\mathcal{L}_{masKD}^{level} = \left\| \frac{Q^{level}(F_{student}^{level})}{\|Q^{level}(F_{student}^{level})\|_2} - \frac{Q^{level}(F_{fusion}^{level})}{\|Q^{level}(F_{fusion}^{level})\|_2} \right\|_2 \tag{7}$$

$$\mathcal{L}_{masKD} = \sum_{level \in \{low, high\}} \mathcal{L}_{masKD}^{level}, \quad \mathcal{L}_{overall}^{student} = \mathcal{L}_{task}^{student} + \lambda\mathcal{L}_{masKD} \tag{8}$$

# 4 Experiments

## 4.1 Experimental Setup

The versatility of the KD framework is verified by adopting both camera and LiDAR student modalities, and by evaluating on two 3D perception tasks: 3D object detection and BEV map segmentation. Notably, our framework can be easily extended to support additional student modalities such as radar and event-based cameras, and to support other downstream tasks, including 3D object tracking and motion prediction.

**Dataset & Evaluation Metrics** The training and evaluation are conducted using the nuScenes dataset[4], a large-scale dataset under the CC BY-NC-SA 4.0 license comprising 1,000 driving sequences (700/150/150 for train/val/test). This dataset provides diverse annotations and sensor data, featuring six monocular camera images along with a 32-beam LiDAR system, making it ideal for assessing our method across various tasks and modalities. We employ evaluation metrics that are widely adopted by state-of-the-art methods. For 3D object detection, we adopt the mean Average Precision (mAP) along with the nuScenes detection score (NDS), the official evaluation metrics of the nuScenes dataset [4]. For BEV map segmentation, following BEVFusion [28] we use the class-averaged mean Intersection-over-Union (mIoU) across six background classes (drivable space, pedestrian crossing, walkway, stop line, car-parking area, and lane divider) as the evaluation metrics.

**Models & Evaluation Configuration** We use MMDetection3D [10] and MMRazor [11] under the Apache License 2.0 to implement the fusion module and the knowledge distillation framework. We replace BEVFusion's [28] fusion module with our Modality-general Fusion Module to serve as the teacher, keeping all other settings identical to BEVFusion. To verify the versatility of our framework, we conduct KD on various representative student models, including the CenterPoint[55] LiDAR student model, BEVDet[16] with a ResNet-50 image backbone, and BEVFormer-S[25] (without temporal fusion) as the camera student models. Additionally, we conduct cross-modal KD experiments on BEVDet4D-Depth[15] student model, which adopts the long-term temporal fusion operation. We maintain the same model architectures and data pipeline as original student models. For the segmentation task, we simply replace the detection head with the BEVFusion [28] segmentation head for binary segmentation across all classes. The teacher model utilizes a complex DETR detection head [2], contrasting with the student models' simpler dense detection heads. This helps us assess the KD method's robustness against heterogeneity in model architecture, besides input modality. Detailed experiment settings can be referred to appendix. B.

## 4.2 Comparison with the State-of-the-Arts

Table 1 presents the comparative results. We choose representative and influential single-modal algorithms as our baseline student models, including CenterPoint [55] for LiDAR, BEVDet-R50 [16], and BEVFormer-S [25] for cameras. In 3D object detection, our framework achieves significant improvements over these baseline models. Our simple LiDAR student performs comparably to those of single-modal state-of-the-art methods, such as TransFusion-L [2] and BEVFusion-L [28], which employ a more complex DETR detection head, thereby narrowing the performance gap with more sophisticated multi-modal fusion methods. Additionally, our simple BEVDet-R50 student outperforms FCOS3D [44] and BEVFusion-C [28], which use heavier image backbones. Our method also exceeds the performance of state-of-the-art cross-modal KD methods, including UniDistill [68] and BEVDistill [7], both of which involve elaborate response KD. This underscores the effectiveness of our feature distillation approach, supported by a superior teacher model and learned spatial masks. For BEV map segmentation, our method also shows significant improvements compared to the baselines, validating the task-independence of our integrated fusion and cross-modal KD framework. Additionally, based on the performance comparison with VCD [17] in the 3D object detection task, we observe that while our NDS is slightly lower than that of VCD, this may be due to VCDs use of fine-grained trajectory-based distillation, which provides greater advantages in predicting object motion and velocity. Nevertheless, our method achieves comparable results to VCD in terms of mAP, demonstrating that the multi-sweep LiDAR information in the teacher model used by VeXKD significantly enhances the student's localization capabilities.

Table 1: **Performance of VeXKD on nuScenes for 3D object detection and BEV map segmentation tasks.** "L" and "C" denote the LiDAR and the Camera modality, respectively. "L+C" denotes multi-modal fusion model. "L+C → C", "L → C", and "L+C → L" represent the knowledge distillation from the teacher model to the respective single-modal student."+" indicates the addition of cross-modal KD methods to the above student models. The FPS results are evaluated on GTX 4090 GPU with batch size of one. "*" denotes our re-implementation results. No test-time augmentation is applied during testing.

| Method | Modality | GFLOPs | FPS | nuScenes test | | nuScenes val | | |
|---|---|---|---|---|---|---|---|---|
| | | | | mAP | NDS | mAP | NDS | mIoU |
| TransFusion[2] | L+C | 972 | 1.8 | 68.9 | 71.6 | 67.5 | 71.3 | – |
| BEVFusion[28] | L+C | 507 | 2.3 | 70.2 | 72.9 | 68.5 | 71.4 | 62.9 |
| TransFusion-L[2] | L | 340 | 5.8 | 65.5 | 70.2 | 65.1 | 70.1 | – |
| BEVFusion-L[28] | L | 322 | 5.4 | – | – | 64.7 | 69.3 | 48.6 |
| CenterPoint[55] | L | 308 | 11.3 | 60.3 | 67.3 | 57.4 | 65.6 | 48.6 |
| +S2M2-SSD [64] | L+C → L | 308 | 11.3 | 63.6 | 69.6 | – | – | – |
| +Unidistill [68] | L+C → L | 308 | 11.3 | 63.9 | 70.1 | 59.7 | 67.5 | – |
| +VeXKD(Ours) | L+C → L | 308 | 11.3 | **65.1** | **70.5** | **64.2** | **69.6** | **52.1** |
| FCOS3d[44] | C | 2008 | 1.7 | 34.3 | 41.5 | 29.5 | 37.2 | – |
| BEVDet-Tiny[16] | C | 370 | 6.3 | – | – | 33.3 | 41.0 | 56.8* |
| BEVDet-R50[16] | C | 184 | 14.0 | 28.9 | 38.4 | 28.6 | 37.2 | 56.4* |
| +Unidistill[68] | L+C → C | 184 | 14.0 | 29.6 | 39.3 | – | – | – |
| +VeXKD(Ours) | L+C → C | 184 | 14.0 | **35.8** | **42.6** | **34.7** | **40.6** | **60.7** |
| BEVFormer-S[25] | C | 1152 | 2.6 | 40.9 | 46.2 | 37.5 | 44.8 | 61.8* |
| +Unidistill[68] | L+C → C | 1152 | 2.6 | – | – | 37.7* | 45.5* | – |
| +BEVDistill[7] | L → C | 1152 | 2.6 | – | – | 38.6 | 45.7 | – |
| +VeXKD(Ours) | L+C → C | 1152 | 2.6 | **42.5** | **48.3** | **41.2** | **47.7** | **64.2** |
| BEVDet4D-Depth[15] | C | 220 | 12.3 | – | – | 39.4 | 51.5 | 61.6* |
| +VCD[17] | L+C | 220 | 12.3 | – | – | 42.6 | **54.0** | – |
| +VeXKD(Ours) | L+C | 220 | 12.3 | – | – | **42.8** | 53.5 | **63.5** |

Furthermore, the comparison of giga floating-point operations (GFLOPs) and inference time clearly illustrates the performance and real-time trade-off introduced by cross-modal KD methods, particularly with our VeXKD on the student model.

### 4.3 Ablation Studies

In this section, we validate the effectiveness of each module. Given the diversity of modalities and tasks applicable to our method, some ablation studies are conducted on specific modality-task combinations due to computational constraints.

**Effectiveness of Each Proposed Module** In this paper, we introduce three modules: the Modality-General Fusion Module (MGFM) and Masked Feature Distillation modules for both **L**ow- and **H**igh-level BEV features (L-MFD and H-MFD). Configurations without MGFM use the original BEVFusion as teachers. We conduct experiments on CenterPoint detection and BEVDet-R50 segmentation students to evaluate the impact of these modules on different student modalities and tasks. Results in Table 2 indicate each module positively contributes to the effectiveness of the KD, with MGFM showing the most substantial impact. This confirms the previously overlooked critical role of the teacher model in cross-modal KD settings.

**Effectiveness of BEV Query Guided Mask Generation Network** To validate the effectiveness of the learned spatial mask, comparisons are made with state-of-the-art masking methods on the CenterPoint student model: complete feature mimic (all 1's mask) [5], key point sampling [68], Gaussian masks centered on ground truth [7], and masks derived from teacher's normalized activation map statistics [57]. In map segmentation, key point sampling and Gaussian masks, which rely on ground truth, are inapplicable. Table 3 shows minimal improvements from complete fea-

Table 2: **Ablation study of three proposed algorithm components on nuScenes val.** MGFM denotes the Modality-General Fusion Module, L-MFD denotes Low-level Masked Feature Distillation, and H-MFD denotes High-level Masked Feature Distillation. Performance metrics include mAP/NDS for detection and mIoU for the segmentation task.

| | Component | | | Student Modality & Task | |
| Setting | MGFM | L-MFD | H-MFD | LiDAR Detection | Camera Segmentation |
|---|---|---|---|---|---|
| 1 | | | | 57.4/65.6 | 56.4 |
| 2 | | ✓ | | 59.2/66.8 (+1.8/1.2) | 57.6 (+1.2) |
| 3 | | | ✓ | 58.0/66.1 (+0.6/0.5) | 57.9 (+1.5) |
| 4 | | ✓ | ✓ | 59.8/66.9 (+2.4/1.3) | 58.3 (+1.9) |
| 5 | ✓ | ✓ | | 62.8/68.9 (+5.4/3.4) | 59.2 (+2.8) |
| 6 | ✓ | | ✓ | 61.9/68.5 (+4.5/2.9) | 59.6 (+3.2) |
| 7 | ✓ | ✓ | ✓ | 64.2/69.6 (+6.8/4.0) | 60.7 (+4.3) |

ture mimic, underscoring the necessity for feature filtering through masking. Gaussian masks and key point sampling focus on limited foreground points for all levels of features, result in underutilized feature distillation. Using normalized activation statistics as masks, assuming that regions with higher teacher activation contain critical information, shows a correlation but not equivalence. Meanwhile, the varying performance across tasks underscores the necessity of learning specialized masks for a versatile KD framework. Additionally, comparisons between Mask Generation with Randomly Initialized Queries (MGTIQ) and Mask Generation with Learned Queries (MGLQ) demonstrate the benefits of using fusion byproducts in mask learning.

Table 3: **Ablation study of different mask selection methods in feature distillation on nuScenes val.**

| | Student Modality & Task | |
| Method | LiDAR Detection | LiDAR Segementation |
|---|---|---|
| No feature distillation | 57.4/65.6 | 48.6 |
| Feature distillation on entire feature maps[5] | 58.7/66.4 | 50.5 |
| Feature distillation on foreground key points[68] | 60.3/67.0 | – |
| Feature distillation masked by Gaussian[7] | 60.8/67.3 | – |
| Feature distillation on activation value[57] | 61.4/67.7 | 50.9 |
| Mask Generation with Randomly Initialized Query | 63.7/69.1 | 51.6 |
| Mask Generation with Learned Query from fusion | 64.2/69.6 | 52.1 |

**Effectiveness of Attention Transfer** We use attention transfer (ATTN) [56] to compute the feature distillation loss, as it helps alleviate the channel dimension heterogeneity from differing architectures. Table 4 presents a comparison of results using attention transfer versus traditional loss L1, L2, and Smooth L1 on CenterPoint student model, each implemented with a simple convolutional adaptive layer. Although employing MGFM and learned masks renders the choice of specific distillation loss less critical, attention transfer consistently outperforms other losses across tasks.

Table 4: **Ablation study of different loss functions in feature distillation on nuScenes val.** The choice of specific distillation loss is less critical while attention transfer performs better.

| | Student Modality & Task | |
| Method | LiDAR Det. | LiDAR Seg. |
|---|---|---|
| No KD | 57.4/65.6 | 48.6 |
| L1 | 62.8/69.0 | 51.3 |
| L2 | 63.7/69.2 | 51.6 |
| Smooth-L1 | 63.5/69.3 | 51.7 |
| ATTN[56] | 64.2/69.6 | 52.1 |

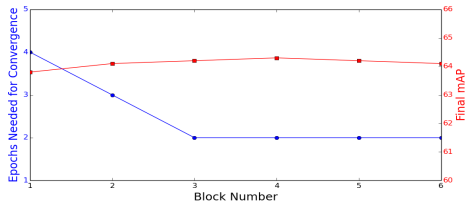

Figure 5: Ablation study of the blocks number in the mask generation network on nuScenes val.

**Influence of Mask Generation Network Architecture** Our mask generation network consists of stacked transformer blocks. We examine how the number of blocks affects mask learning convergence [†] speed and KD results on CenterPoint student model. Figure 5 shows that increasing the block count beyond two does not significantly enhance KD performance. However, employing three or more blocks accelerates convergence. To balance learning speed and performance, we opt for a block count of three.

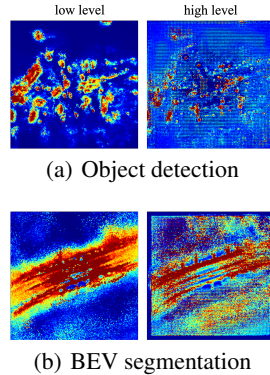

(a) Object detection

(b) BEV segmentation

Figure 6: **Visualization of learned spatial mask.** Our mask generation network can produce spatial masks for features at different levels, tailored for various 3D perception tasks.

### 4.4 Qualitative Results

Figure 6 shows spatial masks for different levels and tasks created by our BEV Query Guided Mask Generation module. In detection tasks, the mask emphasizes a wider area around foreground objects, especially in low-level BEV features; in segmentation tasks, it also highlights background features. Figure 7 displays BEV features before and after our fusion module, demonstrating that our fusion module effectively enhances crucial areas with camera contextual information. Figure 8 compares the BEV feature maps before and after KD, illustrating how our method improves camera features with more deterministic depth projection accuracy and accentuates important LiDAR features by correlating point distribution with image textural information.

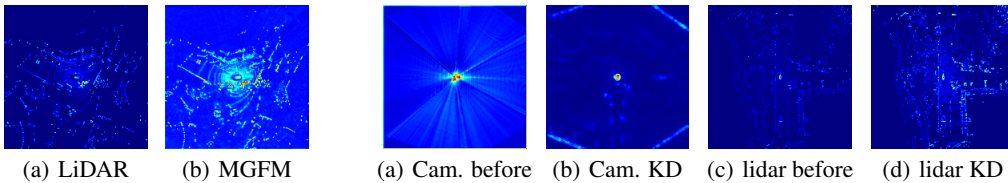

(a) LiDAR  (b) MGFM    (a) Cam. before  (b) Cam. KD  (c) lidar before  (d) lidar KD

Figure 7: **Feature maps before and after MGFM**, showing enhancements in crucial areas with camera textural feature.

Figure 8: **Comparison of feature maps – without vs. with distillation.** KD improves camera feature, offering more deterministic view projection accuracy and accentuating important LiDAR features.

## 5    Conclusion

We propose VeXKD, a simple and versatile framework that integrates cross-modal fusion and knowledge distillation, suitable for various student modalities and 3D perception tasks. Our approach constructs a superior fusion model as the teacher and enhances the effectiveness of cross-modal knowledge distillation. Using a BEV query guided mask generation network, we develop an adaptable feature distillation pipeline that produces spatial masks for features across different tasks and student models, demonstrating its potential for integration with future 3D perception methods. The effectiveness of our framework is confirmed through experiments on the nuScenes dataset, aiming to spur further research into versatile KD frameworks that move beyond model-specific and intricately engineered setups to more universally adaptable approaches.

**Limitations** While our framework is adaptable and has been tested across various modalities and tasks, many scenarios remain untested due to time and computational constraints, such as radar modality or the motion prediction task. The modality-general information utilized in our fusion model has only been shown through experimental outcomes and visualizations. Currently, no studies can theoretically quantify this information. Future research could aim to extend knowledge distillation to more 3D perception tasks and explore the theoretical aspects that influence its efficacy. We utilize the implicit multi-sweep LiDAR information from the teacher model for temporal knowledge transfer to the student model. The comparison with VCD demonstrates the potential of integrating explicit temporal knowledge distillation operations. Developing versatile cross-modal KD frameworks based on explicit temporal knowledge transfer could be a promising future direction.

---

[†]Mask convergence is defined as the condition where the difference in loss for the teacher's downstream task between the post-mask and original feature maps is less than 0.1.

## Acknowledgements

This work was partially supported by the National Natural Science Foundation of China (Grants 62373315, U23A20339, 62373298), the Guangdong Provincial Key Lab of Integrated Communication, Sensing and Computation for Ubiquitous Internet of Things (2023B1212010007), the Guangzhou Basic and Applied Basic Research Project (2023A03J0683), the Guangzhou Municipal Science and Technology Project (2023A03J0011), and the Nansha Key Science and Technology Project (2023ZD006).

## References

[1] Eduardo Arnold, Omar Y Al-Jarrah, Mehrdad Dianati, Saber Fallah, David Oxtoby, and Alex Mouzakitis. A survey on 3d object detection methods for autonomous driving applications. *IEEE Transactions on Intelligent Transportation Systems*, 20(10):3782–3795, 2019.

[2] Xuyang Bai, Zeyu Hu, Xinge Zhu, Qingqiu Huang, Yilun Chen, Hongbo Fu, and Chiew-Lan Tai. Transfusion: Robust lidar-camera fusion for 3d object detection with transformers. In *Proceedings of the IEEE/CVF conference on computer vision and pattern recognition*, pages 1090–1099, 2022.

[3] Cristian Bucilu, Rich Caruana, and Alexandru Niculescu-Mizil. Model compression. In *Proceedings of the 12th ACM SIGKDD international conference on Knowledge discovery and data mining*, pages 535–541, 2006.

[4] Holger Caesar, Varun Bankiti, Alex H Lang, Sourabh Vora, Venice Erin Liong, Qiang Xu, Anush Krishnan, Yu Pan, Giancarlo Baldan, and Oscar Beijbom. nuscenes: A multimodal dataset for autonomous driving. In *Proceedings of the IEEE/CVF conference on computer vision and pattern recognition*, pages 11621–11631, 2020.

[5] Weihan Cao, Yifan Zhang, Jianfei Gao, Anda Cheng, Ke Cheng, and Jian Cheng. Pkd: General distillation framework for object detectors via pearson correlation coefficient. *Advances in Neural Information Processing Systems*, 35:15394–15406, 2022.

[6] Xiaozhi Chen, Huimin Ma, Ji Wan, Bo Li, and Tian Xia. Multi-view 3d object detection network for autonomous driving. In *Proceedings of the IEEE conference on Computer Vision and Pattern Recognition*, pages 1907–1915, 2017.

[7] Zehui Chen, Zhenyu Li, Shiquan Zhang, Liangji Fang, Qinhong Jiang, and Feng Zhao. Bevdistill: Cross-modal bev distillation for multi-view 3d object detection. *arXiv preprint arXiv:2211.09386*, 2022.

[8] Jang Hyun Cho and Bharath Hariharan. On the efficacy of knowledge distillation. In *Proceedings of the IEEE/CVF international conference on computer vision*, pages 4794–4802, 2019.

[9] Zhiyu Chong, Xinzhu Ma, Hong Zhang, Yuxin Yue, Haojie Li, Zhihui Wang, and Wanli Ouyang. Monodistill: Learning spatial features for monocular 3d object detection. *arXiv preprint arXiv:2201.10830*, 2022.

[10] MMDetection3D Contributors. MMDetection3D: OpenMMLab next-generation platform for general 3D object detection. `https://github.com/open-mmlab/mmdetection3d`, 2020.

[11] MMRazor Contributors. Openmmlab model compression toolbox and benchmark. `https://github.com/open-mmlab/mmrazor`, 2021.

[12] Jiajun Deng, Shaoshuai Shi, Peiwei Li, Wengang Zhou, Yanyong Zhang, and Houqiang Li. Voxel r-cnn: Towards high performance voxel-based 3d object detection. In *Proceedings of the AAAI conference on artificial intelligence*, volume 35, pages 1201–1209, 2021.

[13] Jianyuan Guo, Kai Han, Yunhe Wang, Han Wu, Xinghao Chen, Chunjing Xu, and Chang Xu. Distilling object detectors via decoupled features. In *Proceedings of the IEEE/CVF Conference on Computer Vision and Pattern Recognition*, pages 2154–2164, 2021.

[14] Geoffrey Hinton, Oriol Vinyals, and Jeff Dean. Distilling the knowledge in a neural network. *arXiv preprint arXiv:1503.02531*, 2015.

[15] Junjie Huang and Guan Huang. Bevdet4d: Exploit temporal cues in multi-camera 3d object detection. *arXiv preprint arXiv:2203.17054*, 2022.

[16] Junjie Huang, Guan Huang, Zheng Zhu, Yun Ye, and Dalong Du. Bevdet: High-performance multi-camera 3d object detection in bird-eye-view. *arXiv preprint arXiv:2112.11790*, 2021.

[17] Linyan Huang, Zhiqi Li, Chonghao Sima, Wenhai Wang, Jingdong Wang, Yu Qiao, and Hongyang Li. Leveraging vision-centric multi-modal expertise for 3d object detection. *Advances in Neural Information Processing Systems*, 36, 2024.

[18] Tao Huang, Yuan Zhang, Shan You, Fei Wang, Chen Qian, Jian Cao, and Chang Xu. Masked distillation with receptive tokens. *arXiv preprint arXiv:2205.14589*, 2022.

[19] Alex H Lang, Sourabh Vora, Holger Caesar, Lubing Zhou, Jiong Yang, and Oscar Beijbom. Pointpillars: Fast encoders for object detection from point clouds. In *Proceedings of the IEEE/CVF conference on computer vision and pattern recognition*, pages 12697–12705, 2019.

[20] Jianing Li, Ming Lu, Jiaming Liu, Yandong Guo, Yuan Du, Li Du, and Shanghang Zhang. Bev-lgkd: A unified lidar-guided knowledge distillation framework for multi-view bev 3d object detection. *IEEE Transactions on Intelligent Vehicles*, 2023.

[21] Quanquan Li, Shengying Jin, and Junjie Yan. Mimicking very efficient network for object detection. In *Proceedings of the ieee conference on computer vision and pattern recognition*, pages 6356–6364, 2017.

[22] Qi Li, Yue Wang, Yilun Wang, and Hang Zhao. Hdmapnet: An online hd map construction and evaluation framework. In *2022 International Conference on Robotics and Automation (ICRA)*, pages 4628–4634. IEEE, 2022.

[23] Yinhao Li, Zheng Ge, Guanyi Yu, Jinrong Yang, Zengran Wang, Yukang Shi, Jianjian Sun, and Zeming Li. Bevdepth: Acquisition of reliable depth for multi-view 3d object detection. In *Proceedings of the AAAI Conference on Artificial Intelligence*, volume 37, pages 1477–1485, 2023.

[24] Yingwei Li, Adams Wei Yu, Tianjian Meng, Ben Caine, Jiquan Ngiam, Daiyi Peng, Junyang Shen, Yifeng Lu, Denny Zhou, Quoc V Le, et al. Deepfusion: Lidar-camera deep fusion for multi-modal 3d object detection. In *Proceedings of the IEEE/CVF Conference on Computer Vision and Pattern Recognition*, pages 17182–17191, 2022.

[25] Zhiqi Li, Wenhai Wang, Hongyang Li, Enze Xie, Chonghao Sima, Tong Lu, Yu Qiao, and Jifeng Dai. Bevformer: Learning birds-eye-view representation from multi-camera images via spatiotemporal transformers. In *European conference on computer vision*, pages 1–18. Springer, 2022.

[26] Ming Liang, Bin Yang, Shenlong Wang, and Raquel Urtasun. Deep continuous fusion for multi-sensor 3d object detection. In *Proceedings of the European conference on computer vision (ECCV)*, pages 641–656, 2018.

[27] Yingfei Liu, Tiancai Wang, Xiangyu Zhang, and Jian Sun. Petr: Position embedding transformation for multi-view 3d object detection. In *European Conference on Computer Vision*, pages 531–548. Springer, 2022.

[28] Zhijian Liu, Haotian Tang, Alexander Amini, Xinyu Yang, Huizi Mao, Daniela L Rus, and Song Han. Bevfusion: Multi-task multi-sensor fusion with unified bird's-eye view representation. In *2023 IEEE international conference on robotics and automation (ICRA)*, pages 2774–2781. IEEE, 2023.

[29] Ilya Loshchilov and Frank Hutter. Decoupled weight decay regularization, 2019.

[30] Qianhui Luo, Huifang Ma, Li Tang, Yue Wang, and Rong Xiong. 3d-ssd: Learning hierarchical features from rgb-d images for amodal 3d object detection. *Neurocomputing*, 378:364–374, 2020.

[31] Ishan Misra, Rohit Girdhar, and Armand Joulin. An end-to-end transformer model for 3d object detection. In *Proceedings of the IEEE/CVF International Conference on Computer Vision*, pages 2906–2917, 2021.

[32] Jonah Philion and Sanja Fidler. Lift, splat, shoot: Encoding images from arbitrary camera rigs by implicitly unprojecting to 3d. In *Computer Vision–ECCV 2020: 16th European Conference, Glasgow, UK, August 23–28, 2020, Proceedings, Part XIV 16*, pages 194–210. Springer, 2020.

[33] Danila Rukhovich, Anna Vorontsova, and Anton Konushin. Fcaf3d: Fully convolutional anchor-free 3d object detection. In *European Conference on Computer Vision*, pages 477–493. Springer, 2022.

[34] Danila Rukhovich, Anna Vorontsova, and Anton Konushin. Imvoxelnet: Image to voxels projection for monocular and multi-view general-purpose 3d object detection. In *Proceedings of the IEEE/CVF Winter Conference on Applications of Computer Vision*, pages 2397–2406, 2022.

[35] Ruoyu Sun, Fuhui Tang, Xiaopeng Zhang, Hongkai Xiong, and Qi Tian. Distilling object detectors with task adaptive regularization. *arXiv preprint arXiv:2006.13108*, 2020.

[36] Fida Mohammad Thoker and Juergen Gall. Cross-modal knowledge distillation for action recognition. In *2019 IEEE International Conference on Image Processing (ICIP)*, pages 6–10. IEEE, 2019.

[37] Zhi Tian, Chunhua Shen, Hao Chen, and Tong He. Fcos: A simple and strong anchor-free object detector. *IEEE Transactions on Pattern Analysis and Machine Intelligence*, 44(4):1922–1933, 2020.

[38] Gustavo Velasco-Hernandez, John Barry, Joseph Walsh, et al. Autonomous driving architectures, perception and data fusion: A review. In *2020 IEEE 16th International Conference on Intelligent Computer Communication and Processing (ICCP)*, pages 315–321. IEEE, 2020.

[39] Sourabh Vora, Alex H Lang, Bassam Helou, and Oscar Beijbom. Pointpainting: Sequential fusion for 3d object detection. In *Proceedings of the IEEE/CVF conference on computer vision and pattern recognition*, pages 4604–4612, 2020.

[40] Chunwei Wang, Chao Ma, Ming Zhu, and Xiaokang Yang. Pointaugmenting: Cross-modal augmentation for 3d object detection. In *Proceedings of the IEEE/CVF Conference on Computer Vision and Pattern Recognition*, pages 11794–11803, 2021.

[41] Guo-Hua Wang, Yifan Ge, and Jianxin Wu. Distilling knowledge by mimicking features. *IEEE Transactions on Pattern Analysis and Machine Intelligence*, 44(11):8183–8195, 2021.

[42] Jingdong Wang, Zhen Wei, Ting Zhang, and Wenjun Zeng. Deeply-fused nets. *arXiv preprint arXiv:1605.07716*, 2016.

[43] Tao Wang, Li Yuan, Xiaopeng Zhang, and Jiashi Feng. Distilling object detectors with fine-grained feature imitation. In *Proceedings of the IEEE/CVF Conference on Computer Vision and Pattern Recognition*, pages 4933–4942, 2019.

[44] Tai Wang, Xinge Zhu, Jiangmiao Pang, and Dahua Lin. Fcos3d: Fully convolutional one-stage monocular 3d object detection. In *Proceedings of the IEEE/CVF International Conference on Computer Vision*, pages 913–922, 2021.

[45] Yue Wang, Vitor Campagnolo Guizilini, Tianyuan Zhang, Yilun Wang, Hang Zhao, and Justin Solomon. Detr3d: 3d object detection from multi-view images via 3d-to-2d queries. In *Conference on Robot Learning*, pages 180–191. PMLR, 2022.

[46] Yue Wang, Yongbin Sun, Ziwei Liu, Sanjay E Sarma, Michael M Bronstein, and Justin M Solomon. Dynamic graph cnn for learning on point clouds. *ACM Transactions on Graphics (tog)*, 38(5):1–12, 2019.

[47] Zeyu Wang, Dingwen Li, Chenxu Luo, Cihang Xie, and Xiaodong Yang. Distillbev: Boosting multi-camera 3d object detection with cross-modal knowledge distillation. In *Proceedings of the IEEE/CVF International Conference on Computer Vision (ICCV)*, pages 8637–8646, October 2023.

[48] Yutian Wu, Yueyu Wang, Shuwei Zhang, and Harutoshi Ogai. Deep 3d object detection networks using lidar data: A review. *IEEE Sensors Journal*, 21(2):1152–1171, 2020.

[49] Mutian Xu, Runyu Ding, Hengshuang Zhao, and Xiaojuan Qi. Paconv: Position adaptive convolution with dynamic kernel assembling on point clouds. In *Proceedings of the IEEE/CVF Conference on Computer Vision and Pattern Recognition*, pages 3173–3182, 2021.

[50] Zihui Xue, Zhengqi Gao, Sucheng Ren, and Hang Zhao. The modality focusing hypothesis: Towards understanding crossmodal knowledge distillation. *arXiv preprint arXiv:2206.06487*, 2022.

[51] Yan Yan, Yuxing Mao, and Bo Li. Second: Sparsely embedded convolutional detection. *Sensors*, 18(10):3337, 2018.

[52] Bin Yang, Wenjie Luo, and Raquel Urtasun. Pixor: Real-time 3d object detection from point clouds. In *Proceedings of the IEEE conference on Computer Vision and Pattern Recognition*, pages 7652–7660, 2018.

[53] Chenyu Yang, Yuntao Chen, Hao Tian, Chenxin Tao, Xizhou Zhu, Zhaoxiang Zhang, Gao Huang, Hongyang Li, Yu Qiao, Lewei Lu, et al. Bevformer v2: Adapting modern image backbones to bird's-eye-view recognition via perspective supervision. In *Proceedings of the IEEE/CVF Conference on Computer Vision and Pattern Recognition*, pages 17830–17839, 2023.

[54] Zhendong Yang, Zhe Li, Mingqi Shao, Dachuan Shi, Zehuan Yuan, and Chun Yuan. Masked generative distillation. In *European Conference on Computer Vision*, pages 53–69. Springer, 2022.

[55] Tianwei Yin, Xingyi Zhou, and Philipp Krahenbuhl. Center-based 3d object detection and tracking. In *Proceedings of the IEEE/CVF conference on computer vision and pattern recognition*, pages 11784–11793, 2021.

[56] Sergey Zagoruyko and Nikos Komodakis. Paying more attention to attention: Improving the performance of convolutional neural networks via attention transfer. *arXiv preprint arXiv:1612.03928*, 2016.

[57] Linfeng Zhang and Kaisheng Ma. Improve object detection with feature-based knowledge distillation: Towards accurate and efficient detectors. In *International Conference on Learning Representations*, 2020.

[58] Renrui Zhang, Han Qiu, Tai Wang, Ziyu Guo, Ziteng Cui, Yu Qiao, Hongsheng Li, and Peng Gao. Monodetr: Depth-guided transformer for monocular 3d object detection. In *Proceedings of the IEEE/CVF International Conference on Computer Vision*, pages 9155–9166, 2023.

[59] Su Zhang, Chuangao Tang, and Cuntai Guan. Visual-to-eeg cross-modal knowledge distillation for continuous emotion recognition. *Pattern Recognition*, 130:108833, 2022.

[60] Yunpeng Zhang, Wenzhao Zheng, Zheng Zhu, Guan Huang, Jiwen Lu, and Jie Zhou. A simple baseline for multi-camera 3d object detection. In *Proceedings of the AAAI Conference on Artificial Intelligence*, volume 37, pages 3507–3515, 2023.

[61] Borui Zhao, Quan Cui, Renjie Song, Yiyu Qiu, and Jiajun Liang. Decoupled knowledge distillation. In *Proceedings of the IEEE/CVF Conference on computer vision and pattern recognition*, pages 11953–11962, 2022.

[62] Lin Zhao, Hui Zhou, Xinge Zhu, Xiao Song, Hongsheng Li, and Wenbing Tao. Lif-seg: Lidar and camera image fusion for 3d lidar semantic segmentation. *IEEE Transactions on Multimedia*, 2023.

[63] Xiaodong Zhao, Qichao Zhang, Dongbin Zhao, and Zhonghua Pang. Overview of image segmentation and its application on free space detection. In *2018 IEEE 7th Data Driven Control and Learning Systems Conference (DDCLS)*, pages 1164–1169. IEEE, 2018.

[64] Wu Zheng, Mingxuan Hong, Li Jiang, and Chi-Wing Fu. Boosting 3d object detection by simulating multimodality on point clouds. In *Proceedings of the IEEE/CVF Conference on Computer Vision and Pattern Recognition*, pages 13638–13647, 2022.

[65] Wu Zheng, Weiliang Tang, Sijin Chen, Li Jiang, and Chi-Wing Fu. Cia-ssd: Confident iou-aware single-stage object detector from point cloud. In *Proceedings of the AAAI conference on artificial intelligence*, volume 35, pages 3555–3562, 2021.

[66] Wu Zheng, Weiliang Tang, Li Jiang, and Chi-Wing Fu. Se-ssd: Self-ensembling single-stage object detector from point cloud. In *Proceedings of the IEEE/CVF conference on computer vision and pattern recognition*, pages 14494–14503, 2021.

[67] Brady Zhou and Philipp Krähenbühl. Cross-view transformers for real-time map-view semantic segmentation. In *Proceedings of the IEEE/CVF conference on computer vision and pattern recognition*, pages 13760–13769, 2022.

[68] Shengchao Zhou, Weizhou Liu, Chen Hu, Shuchang Zhou, and Chao Ma. Unidistill: A universal cross-modality knowledge distillation framework for 3d object detection in bird's-eye view. In *Proceedings of the IEEE/CVF Conference on Computer Vision and Pattern Recognition*, pages 5116–5125, 2023.

[69] Yin Zhou and Oncel Tuzel. Voxelnet: End-to-end learning for point cloud based 3d object detection. In *Proceedings of the IEEE conference on computer vision and pattern recognition*, pages 4490–4499, 2018.

[70] Benjin Zhu, Zhengkai Jiang, Xiangxin Zhou, Zeming Li, and Gang Yu. Class-balanced Grouping and Sampling for Point Cloud 3D Object Detection. *arXiv e-prints*, page arXiv:1908.09492, Aug. 2019.

[71] Xizhou Zhu, Weijie Su, Lewei Lu, Bin Li, Xiaogang Wang, and Jifeng Dai. Deformable detr: Deformable transformers for end-to-end object detection. *arXiv preprint arXiv:2010.04159*, 2020.

[72] Xinge Zhu, Hui Zhou, Tai Wang, Fangzhou Hong, Yuexin Ma, Wei Li, Hongsheng Li, and Dahua Lin. Cylindrical and asymmetrical 3d convolution networks for lidar segmentation. In *Proceedings of the IEEE/CVF conference on computer vision and pattern recognition*, pages 9939–9948, 2021.

# Appendix

## A    Preliminary

In the realm of 3D perception conducted in BEV space, whether through single-modal or multi-modal fusion, the paradigm as depicted in Figure A.1 is commonly adopted. Within this paradigm, multi-view camera-based algorithms utilize an image backbone to extract semantic information from multi-view image inputs $I_{img}^{multi\_view}$. This information is then projected onto BEV space from the image space through forward projection [32, 16] or backward projection [25, 53], resulting in low-level image BEV features $F_{img}^{low}$ as in Eq. A.1:

$$F_{img}^{low} = \text{Projection}\left(\text{ImgBackbone}\left(I_{img}^{multi\_view}\right)\right) \tag{A.1}$$

In the context of LiDAR-based algorithms, benefiting from the inherent geometric accuracy of Li-DAR point clouds, the LiDAR input $I_{lidar}$ can be easily pooled into the BEV (Bird's Eye View) space, regardless of whether voxel-based or pillar-based encoders are used. This process yields low-level LiDAR BEV features, denoted as $F_{lidar}^{low}$ as in Eq. A.2:

$$F_{lidar}^{low} = \text{LiDAREncoder}\left(I_{lidar}\right) \tag{A.2}$$

Meanwhile, the multi-modality fusion pipeline incorporates both $F_{lidar}^{low}$ and $F_{img}^{low}$ as inputs. These are subsequently processed through a fusion module to produce fused low-level features $F_{fusion}^{low}$. Specifically, BEVFusion [28] achieves this by concatenating $F_{img}^{low}$ and $F_{lidar}^{low}$, followed by processing through several layers of simple convolution to obtain $F_{fusion}^{low}$ as in Eq. A.3:

$$F_{fusion}^{low} = \text{FusionModule}\left(F_{img}^{low}, F_{lidar}^{low}\right) \tag{A.3}$$

Regardless of whether it is LiDAR-only, camera-only or even fusion model, the low level BEV features are not suitable for direct 3D perception tasks due to the spatial inaccuracy and low semantic level. Therefore, they are transformed into high-level BEV features $F_{modal}^{high}, modal \in \{lidar, img, fusion\}$ through BEV encoders as in Eq. A.4:

$$F_{modal}^{high} = \text{BEVEncoder}\left(F_{modal}^{low}\right) \quad , \quad modal \in \{camera, lidar, fusion\} \tag{A.4}$$

Subsequently, these high-level BEV features are processed through various task-specific heads to obtain the final results for 3D perception tasks, such as 3D map segmentation or object detection. Such a common paradigm provide the foundation of building a versatile knowledge distillation framework that can be applied to various student modalities and 3D perception tasks.

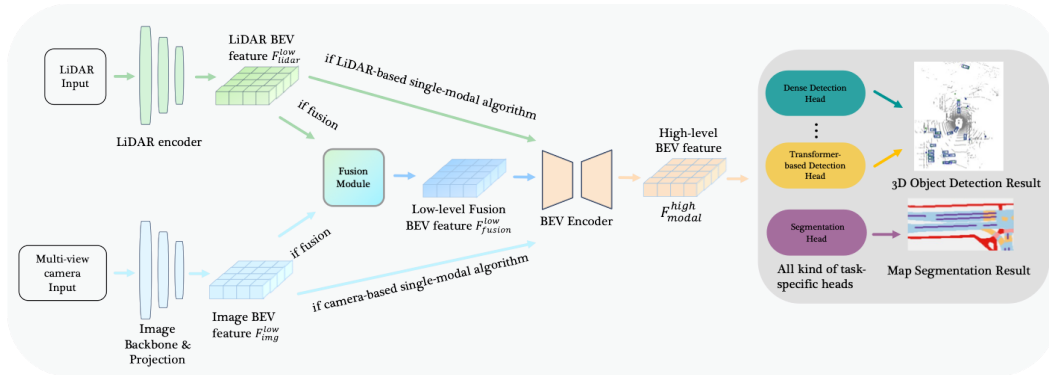

Figure A.1: Common paradigm for BEV-based 3D perception tasks: Despite variations in model architectures, both low-level and high-level BEV features are common to all BEV-based algorithms.

# B  Detailed Training Settings

All experiments are conducted with a batch size of 32, training from scratch on 8 of NVIDIA A800 GPU. By default, each student is trained for 20 epochs using the AdamW optimizer [29] with a weight decay of $10^{-2}$. Throughout the training process of KD, the teacher model remains frozen. The detection range was set to [-54.0, 54.0] for the X and Y axes, and the voxelization resulted in a resolution of (0.075m, 0.075m, 0.2m). Both low-level and high-level feature maps are sized at $180 \times 180$ in height and width. The hyper-parameters applied to the loss terms $\mu$ and $\lambda$ are simply set to 1.0 and 2.0 to balance the scale of losses in our experiment.

**BEVDet-50 Knowledge Distillation Experiment**  Following the original BEVDet [16] experiments, a learning rate of $2 \times 10^{-4}$ is used, with the ResNet-50 image backbone's learning rate reduced by a factor of 0.1 to stabilize training. The input image is cropped to $704 \times 256$, and identical image augmentations are applied to both the student and the teacher to prevent misalignment. For 3D object detection, the CenterPoint detection head is used, and for segmentation, the segmentation head from BEVFusion is employed for comparative experiments before and after KD.

**CenterPoint Knowledge Distillation Experiment**  For the 3D object detection task, a rotate type of Non-Maximum Suppress (NMS) algorithm is used for processing results. For the segmentation task, the CenterPoint head is directly replaced with the segmentation head from BEVFusion for experimental comparison. Following the original settings, CBGS [70] and a one-cycle learning rate policy are employed with an initial learning rate of $1 \times 10^{-4}$.

**BEVFormer-S Knowledge Distillation Experiment**  To mitigate the impact of temporal self-attention in BEVFormer, BEVFormer-S is chosen as the student by adjusting the temporal self-attention into a vanilla self-attention without using historical BEV features. To maintain consistency with the teacher, the resolution of BEVs grid is adjusted from the original 0.512m to 0.6m, thereby reducing the BEV queries size to $180 \times 180$ instead of $200 \times 200$. Training for BEVFormer-S is conducted over 24 epochs, using the same Cosine Annealing learning rate and momentum scheduling as the original. A learning rate of $2 \times 10^{-4}$ is used, with the ResNet-101 image backbone's learning rate reduced by a factor of 0.1 to stabilize training.

**BEVDet4D-Longterm Knowledge distillation Experiment**  This experiment follows the experiment configuration of BEVDet-R50-4DLongterm-Depth-CBGS. The input image size is set to 256Œ704, and the temporal fusion utilizes feature maps from the past 8 frames. ResNet-50 is employed as the image backbone, and CenterHead is used for 3D object detection. Additionally, the depth estimation for the image is supervised using the depth loss proposed in BEVDepth.

# C  More Qualitative Results

**Visualizaiton of BEV features before and after Knowledge Distillation**  As illustrated in Fig. C.1, significant differences are evident in the feature maps of both camera and LiDAR students before and after knowledge distillation (KD). For camera students, KD can serve as a form of depth supervision during the projection process from the perspective view to the BEV (Bird's Eye View). Consequently, compared to the original feature map of BEVDet-R50, which is chaotically distributed across each depth bin, the KD-enhanced feature map contains more accurate and deterministic depth information, benefiting the precision of 3D perception localization. For the LiDAR student, KD helps establish a connection between the point cloud distribution and the camera's textural information. This enhances the visibility of key foreground features, partially compensating for the sparsity of point clouds at greater distances.

**Visualization of detection result before and after Knowledge Distillation**  As previously mentioned, cross-modal KD can act as a regularization mechanism in the projection process for camera-based systems. The detection visualization results, shown in Fig. C.2, demonstrate that KD enhances confidence in depth estimates for camera students, helping to prevent the occurrence of false positives. For LiDAR students, KD facilitates the establishment of a correlation between point distribution and the camera's textural information. This enhancement aids in preventing the misclassification of objects into different categories and reduces false positives. Additionally, KD improves the recognition and accurate classification of objects with fewer points at greater distances.

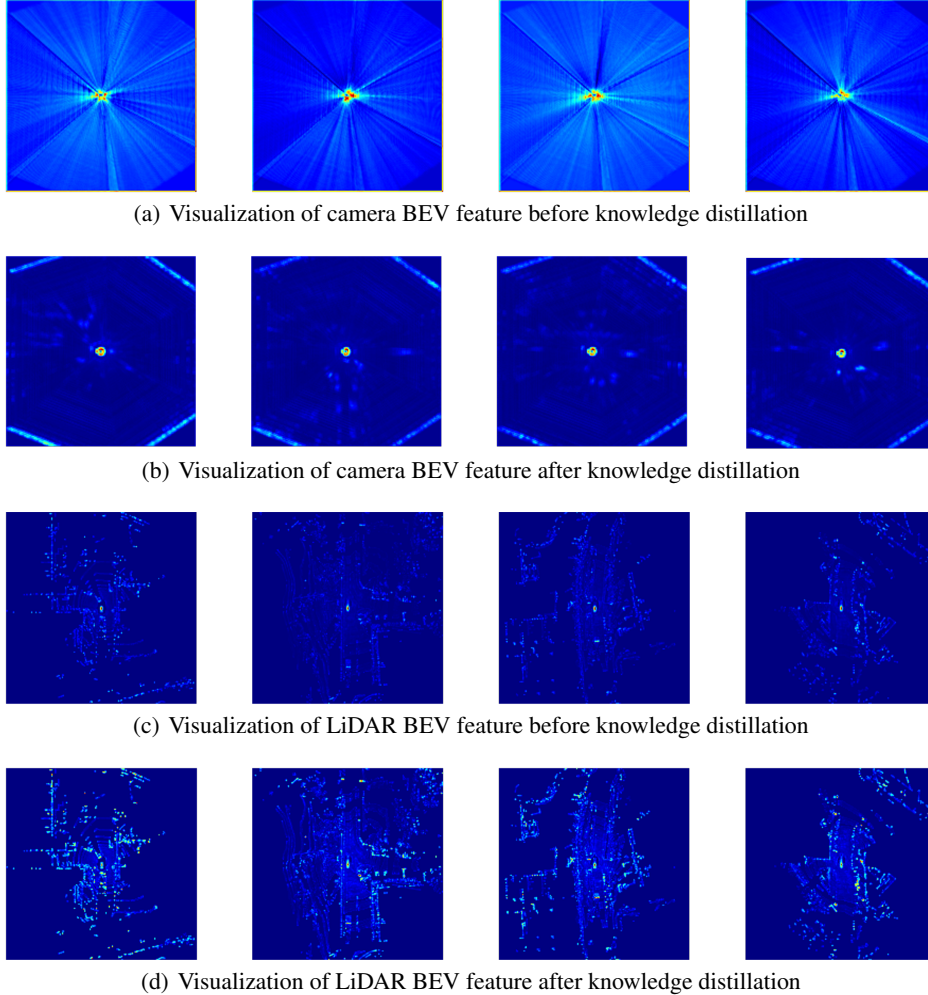

(a) Visualization of camera BEV feature before knowledge distillation

(b) Visualization of camera BEV feature after knowledge distillation

(c) Visualization of LiDAR BEV feature before knowledge distillation

(d) Visualization of LiDAR BEV feature after knowledge distillation

Figure C.1: BEV feature map comparison between KD and no-KD ones

**Visualization of segmentation result before and after Knowledge Distillation** In the task of BEV map segmentation, a primary challenge for the camera modality is the unclear boundary de-lineation and boundary spreading caused by inaccuracies in the view projection process. As shown in Fig. C.3, KD can significantly enhance the accuracy of depth estimation in this projection, facil-itating a more precise reconstruction of the original BEV scene. Conversely, LiDAR models often underperform due to their inherent sparsity and reduced focus on background areas, which typi-cally have fewer reflective points. The framework introduced in this study incorporates a learned spatial mask that increases focus on these background locations, thereby preventing hallucinations near regions with sparse LiDAR points and improving the accuracy of background segmentation. Additionally, the established correlation between camera textural information and point cloud dis-tribution promotes more accurate pixel-wise classification, further enhancing model performance in densely populated nearby areas.

## D Broader Impacts

Effective and real-time 3D perception is crucial for the safety of autonomous vehicles. By integrat-ing cross-modal fusion and knowledge distillation, VeXKD transfers the multi-modal knowledge to improve the accuracy of single-modal student without additional inference time overhead, thus achieving better efficiency and accuracy trade-off. With its versatility, VeXKD can be applied to

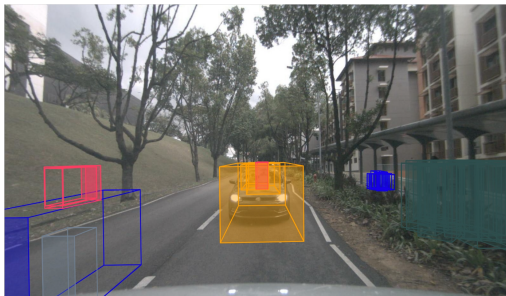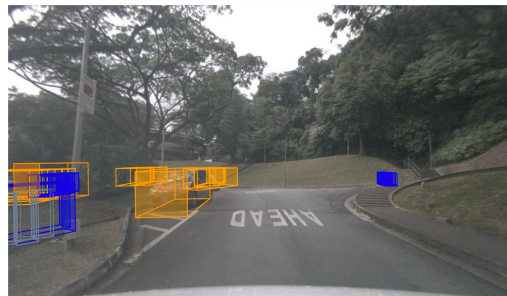

(a) Camera detection result before knowledge distillation

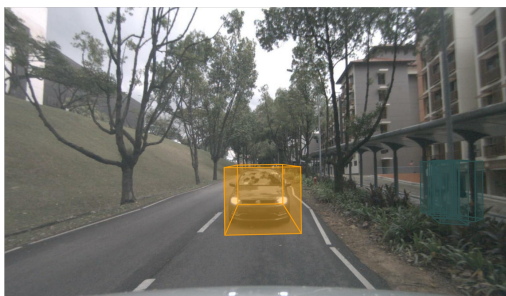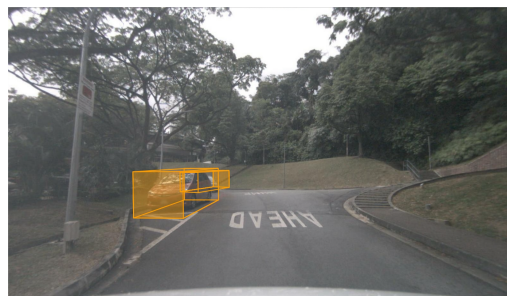

(b) Camera detection result after knowledge distillation

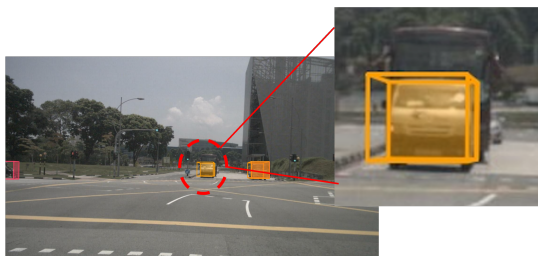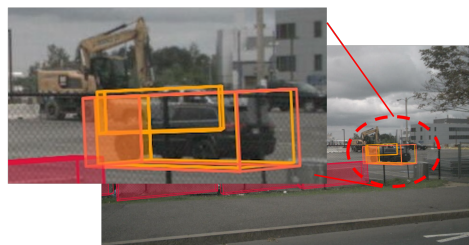

(c) LiDAR detection result before knowledge distillation

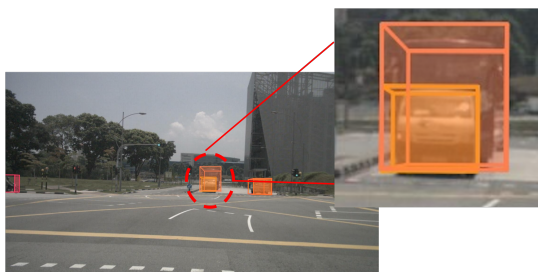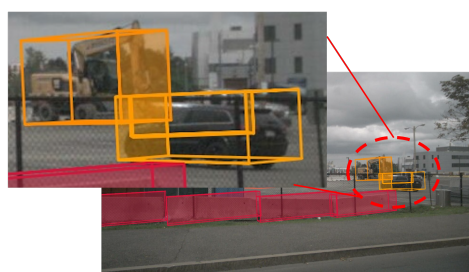

(d) LiDAR detection result after knowledge distillation

Figure C.2: Detection result before and after knowledge distillation for camera and LiDAR students.

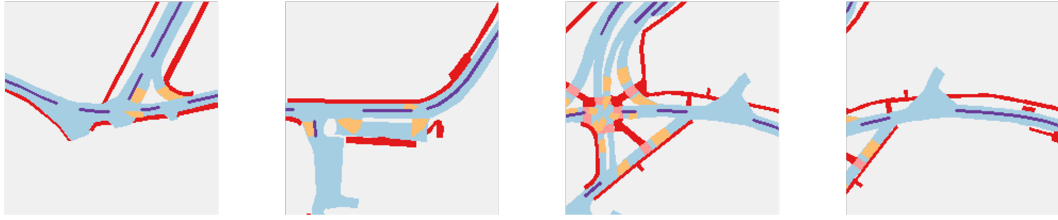

(a) Ground truth segmentation label for camera segmentation

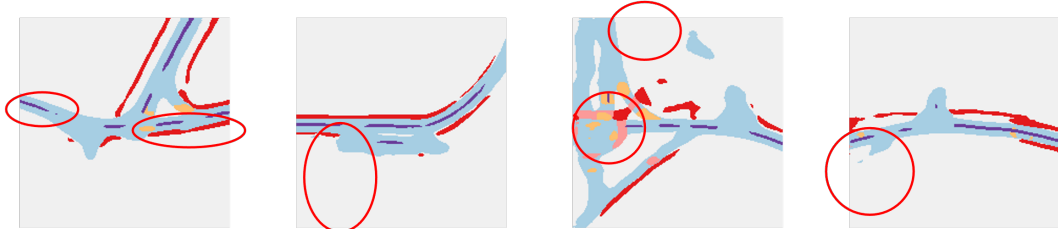

(b) Camera segmentation result before knowledge distillation

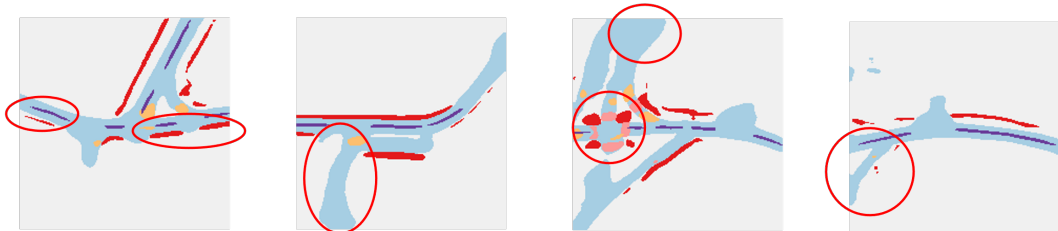

(c) Camera segmentation result after knowledge distillation

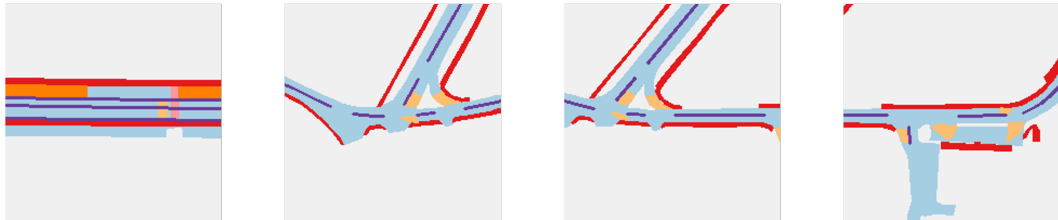

(d) Ground truth segmentation label for LiDAR segmentation

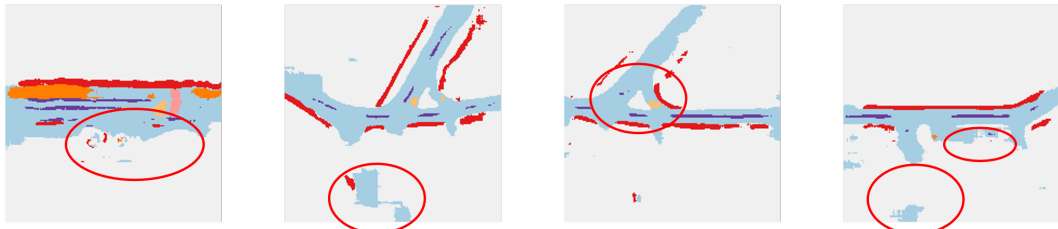

(e) LiDAR segmentation result before knowledge distillation

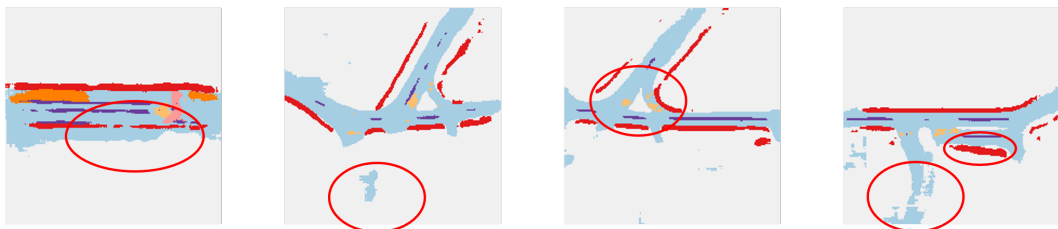

(f) LiDAR segmentation result with knowledge distillation

Figure C.3: Segmentation result before or after knowledge distillation

various student models and downstream 3D perception tasks. It can help pave the way for safe and robust autonomous driving by integrating with the ever-evolving single-modal algorithms.

